# Rethinking No-reference Image Exposure Assessment from Holism to Pixel: Models, Datasets and Benchmarks

**Shuai He**[†,1]    **Shuntian Zheng**[†,2]    **Anlong Ming**[†,1,*]    **Banyu Wu**[1]    **Huadong Ma**[1]

[1] Beijing University of Posts and Telecommunications    [2] University of Warwick

{hs19951021}@bupt.edu.cn

Shuntian.Zheng@warwick.ac.uk {mal, wubanyu, mhd}@bupt.edu.cn

## Abstract

The past decade has witnessed an increasing demand for enhancing image quality through exposure, and as a crucial prerequisite in this endeavor, Image Exposure Assessment (IEA) is now being accorded serious attention. However, IEA encounters two persistent challenges that remain unresolved over the long term: the accuracy and generalizability of No-reference IEA are inadequate for practical applications; the scope of IEA is confined to qualitative and quantitative analysis of the entire image or subimage, such as providing only a score to evaluate the exposure level, thereby lacking intuitive and precise fine-grained evaluation for complex exposure conditions. The objective of this paper is to address the persistent bottleneck challenges from three perspectives: model, dataset, and benchmark. 1) Model-level: we propose a Pixel-level IEA Network (P-IEANet) that utilizes Haar discrete wavelet transform (DWT) to analyze, decompose, and assess exposure from both lightness and structural perspectives, capable of generating pixel-level assessment results under no-reference scenarios. 2) Dataset-level: we elaborately build an exposure-oriented dataset, IEA40K, containing 40K images, covering 17 typical lighting scenarios, 27 devices, and 50+ scenes, with each image densely annotated by more than 10 experts with pixel-level labels. 3) Benchmark-level: we develop a comprehensive benchmark of 19 methods based on IEA40K. Our P-IEANet not only achieves state-of-the-art (SOTA) performance on all metrics but also seamlessly integrates with existing exposure correction and lighting enhancement methods. To our knowledge, this is the first work that explicitly emphasizes assessing complex image exposure problems at a pixel level, providing a significant boost to the IEA and exposure-related community. The code and dataset are available in here.

## 1 Introduction

Exposure, one of the 3A factors (Auto Exposure, Focus and White Balance) in camera technology, plays a crucial role in controlling image quality. Image exposure assessment (IEA) is a prerequisite for improving exposure [1–3]; however, even leading phone and camera manufacturers heavily rely on manual evaluations due to the unavailability and high cost of human raters. Nevertheless, large-scale adoption of manual assessments is impractical. Similar to mainstream AI applications, deep learning and data-driven approaches hold promise as potential solutions to overcome this limitation. Nevertheless, the traditional data-driven IEA paradigm encounters two major challenges:

*1) A Dilemma between Applicability and Practicability:* While full-reference IEA methods deliver satisfactory results [4–9], their applicability in non-preset scenarios is limited due to the general unavailability of reference images. Conversely, no-reference IEA methods, which do not rely on reference information, struggle with natural images distorted by unknown factors. This difficulty

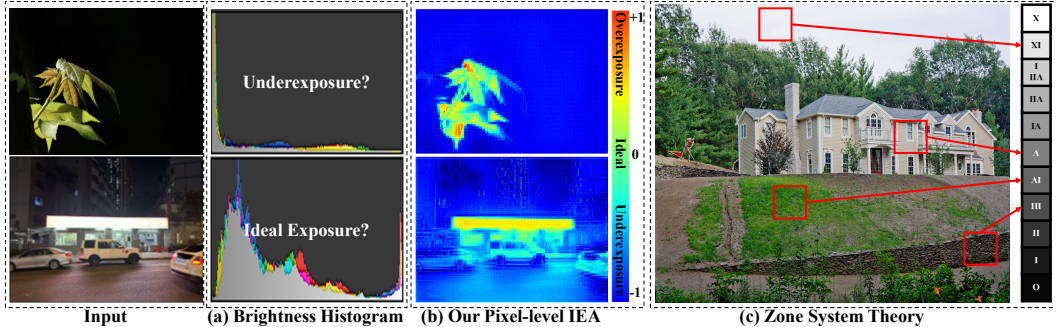

| Input | (a) Brightness Histogram | (b) Our Pixel-level IEA | (c) Zone System Theory |

Figure 1: Comparison between the brightness histogram (a) and our method (b) for assessing image exposure. (b) offers a more intuitive and accurate reflection of the exposure conditions in each area. (c) shows how to determine exposure levels for each area using Adams' zone system theory [14, 15].

arises from the inability to identify specific features for assessing exposure, as the quality prediction problem becomes agnostic to the type of exposure distortion, thereby restricting both performance and practicality [10, 11].

*2) Restricted Generalization Capacity:* Traditional IEA annotation relies on scenario-specific criteria, leading to significant subjectivity across datasets. These datasets typically provide only a holistic quantitative score reflecting the overall exposure condition [12], lacking detailed and fine-grained assessments. Consequently, the subjective and coarse-grained labels introduce restricted generalization capacity into learning-based IEA methods, reducing their adaptability to diverse scenarios and assessment criteria.

How can an ideal method be designed to tackle the aforementioned challenges? The method should primarily address three key issues: firstly, as a no-reference method, it should effectively simulate reference images in non-preset scenarios, functioning like full-reference methods; secondly, it should achieve fine-grained assessments by adapting to diverse high-level evaluation criteria or application scenarios directly or through fine-tuning; finally, the learned features should be decoupled from subjective criteria and aligned with naive exposure features to mitigate narrow inductive biases.

This paper presents P-IEANet, an innovative method that leverages large-scale, pixel-level annotated datasets to delve into the fundamental unit of IEA: pixels. This approach allows us to ***identify exposure issues with unprecedented precision and to handle IEA tasks of varying granularity beyond the pixel level***, without being influenced by subjective criteria. Grounded in the well-established theory that the power spectrum of natural images is a function of frequency, represented as $1/f^\gamma$ where $\gamma$ varies slightly at specific frequencies [11, 13], we leverage this insight to analyze exposure characteristics in specific frequency domains for improved adaptability across varying criteria and scenarios. Through employing the dedicated Haar Discrete Wavelet Transform (DWT), P-IEANet decomposes the original image to criteria-agnostic frequency features, thus avoiding narrow inductive biases. Additionally, with pixel-level supervision, P-IEANet enables ideal exposure reconstruction from frequency space, effectively creating reference images for further analysis.

Our contributions are summarized as follows:

- To our knowledge, this is the first work to implement a pixel-level evaluation paradigm in IEA. It enhances the generalizability and accuracy of no-reference IEA tasks, while effectively addressing challenges associated with reusing underlying data and architectures.

- We present the P-IEANet, showcasing that pixel-level IEA can be decomposed into criteria-agnostic lightness and structure information via the dedicated Haar DWT. This design enables efficient execution of pixel-level IEA while minimizing parameter usage.

- To convincingly validate our method, we have developed a dataset exclusively tailored for IEA, called IEA40K. This dataset specifically focuses on exposure and comprises 40,000 of images with the most comprehensive annotations to date, including pixel-level annotations.

- Building upon IEA40K, we have evaluated 19 baselines, establishing our benchmark as the most comprehensive to date for IEA. Our work not only achieves SOTA performance but

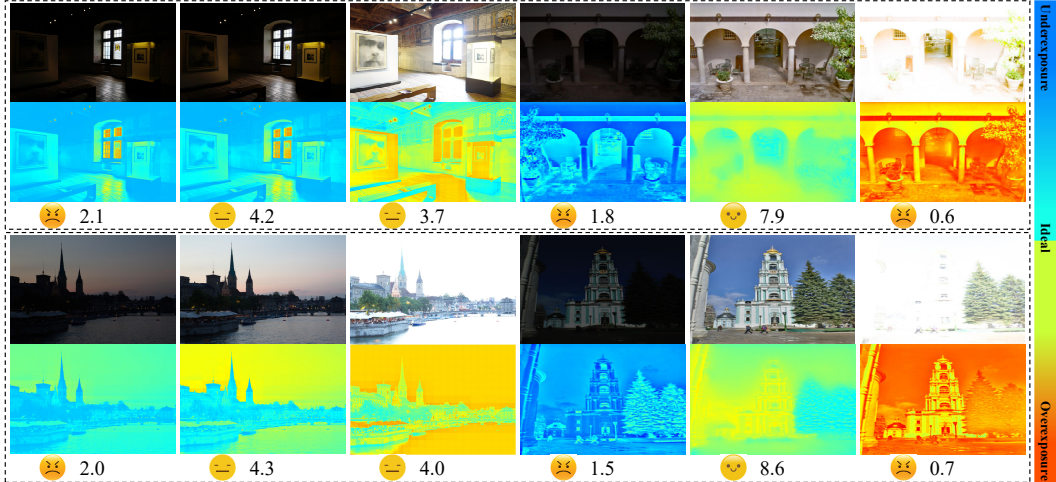

Figure 2: Images visualized along with their corresponding pixel-level (heat map of exposure residual) and holistic IEA results by P-IEANet (where a higher score shows more visually pleasing exposure).

also serves as a pivotal catalyst, offering the community a new roadmap to explore further solutions for IEA.

## 2    Related Work

**No-reference IEA Methods and Datasets.** Previous studies on no-reference IEA can be broadly classified into two primary categories: *1) Statistical-feature based methods*. Datta *et al.* [4] utilized average pixel intensity to evaluate light usage. Liu *et al.*[5] explored image brightness histograms, and Hanmandlu *et al.*[7] developed indicators based on the image brightness histogram for crucial auto-exposure control. Rahman *et al.* [8] and Lu *et al.* [9] adopted information entropy as a criterion for exposure evaluation. In some auto-exposure work, [16–19] incorporated gradient information to determine the ideal exposure settings for cameras, suggesting that maximum information entropy indicates ideal image exposure. Efimov *et al.* [20] and Dong *et al.* [21] proposed subdividing images into blocks for individual assessment, classifying each based on its brightness histogram into categories. *2) Data-driven methods.* The increasingly popular methods [12, 22–24] utilize human-labeled datasets to develop scenario-specific features.

However, when it comes to *statistical-feature based methods*, manual features often assume only one type of distortion, which is problematic in complex situations where overexposure and underexposure coexist [11, 25]. On the other hand, *data-driven methods* become less effective when assessment criteria or application scenarios change; moreover, the holistic scores from these datasets lack the detailed supervisory information required for the precision and granularity demanded in applications.

**Pixel-level Tasks.** In conventional terms, the concept of "exposure" refers to not only exposure time but also two other parameters (aperture and ISO, referred to [26]). Rather than being characterized as a global attribute of the image, the parameters would be more appropriate to be described as a global attribute associated with the camera for capturing the image. However, according to the claim made by the classical photographic theory (Adams' theory) [27] that "The exposure time is the same for all elements, but the image exposure varies with the luminance of each subject element," the coarse global camera exposure attribute fails to match each subject element in an image, potentially resulting in some subject elements being under-exposed and others being over-exposed. Given this, **in the context of evaluating images, the term "exposure" is no longer a global attribute**, as referred to [27] that "Any scene of photographic interest contains elements of different luminance; consequently, **the 'exposure' actually is many different exposures.**" Therefore, the pixel-level IEA is highly desired.

However, to our knowledge, there are currently no pixel-level IEA methods, despite advancements in related visual tasks such as semantic understanding and fine-grained analysis. For instance, DiffuMask [28] exploits powerful zero-shot text-to-image generative models to provide pixel-level

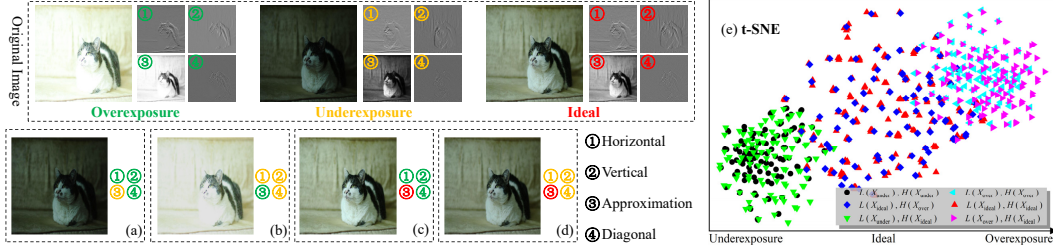

Figure 3: After Haar DWT decomposes an image, swapping its low-frequency component (③) with the high-frequency components (①②④) of the same image under different exposures produces visually similar results (a-d) as well as similar t-SNE features (e).

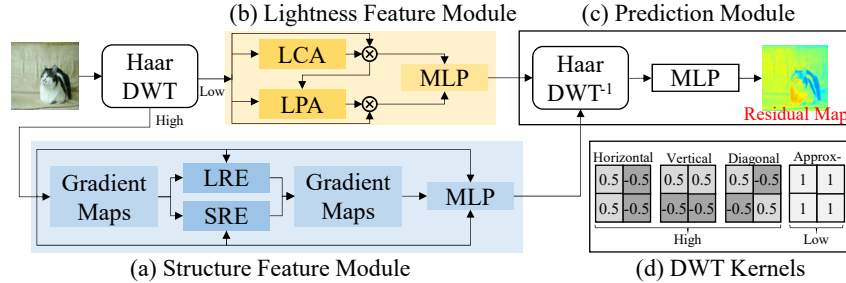

Figure 4: Pipeline of P-IEANet. The original image is decomposed into low/high-frequency components via Haar DWT. Subsequently, the Structure Feature Module analyzes the high-frequency components with gradient maps to extract structural features (*cf.* Sec. 3.1). Simultaneously, the Lightness Feature Module handles the low-frequency component and extracts lightness features by attention mechanisms (*cf.* Sec. 3.2). These features are then composed through Haar inverse DWT for pixel-level IEA. Dedicated convolution kernels (d) are employed to facilitate the DWT process.

segmentation annotations across diverse classes. Similarly, PixelLM [29] leverages GPT4V to produce 246,000 pixel-level question-answer pairs, enhancing its capabilities in pixel-level reasoning and comprehension.

However, directly applying these methods to IEA tasks is very challenging due to specific requirements in data collection and annotation processes, which entail avoiding selection bias, accurately aligning images, obtaining ideal references, and providing detailed annotations (*cf.* Sec. 4).

## 3  Architecture of P-IEANet

**Preliminaries.** Images captured with incorrect exposure settings often suffer from visual problems, including lightness and structure distortions [30–32]. For instance, overexposed images exhibit unnatural artifacts, inconsistencies in exposure blending, and blurred structural details. This paper demonstrates the potential of the Haar DWT for analyzing IEA issues. The mathematical representation of the Haar DWT can be formulated as follows:

$$DWT(L, H) = \frac{1}{\sqrt{2^m}} \sum_k f(k)(\phi\left(\frac{n - k2^m}{2^m}\right), \psi\left(\frac{n - k2^m}{2^m}\right)). \quad (1)$$

The mother wavelet $\phi$ and $\psi$ of the Haar DWT can decompose an image into ***low*** frequency components $L$ (approximation coefficients), and ***high*** frequency components $H$ (coefficients in horizontal, vertical, and diagonal directions), as shown in Fig. 3. We label the underexposed, overexposed, and ideal images as $X_{over}$, $X_{under}$, and $X_{ideal}$, respectively. Their corresponding Haar DWT representations in frequency are denoted as $DWT(L(X_{over}), H(X_{over}))$, $DWT(L(X_{under}), H(X_{under}))$ and $DWT(L(X_{ideal}), H(X_{ideal}))$, respectively.

For IEA tasks, we examine whether the low-frequency and high-frequency components correspond to the frequency-domain representations of lightness and structure, respectively. The images obtained by reversing these components, such as $DWT^{-1}(L(X_{ideal}), H(X_{under}))$ and

$DWT^{-1}(L(X_{ideal}), H(X_{over}))$, show an exposure close to $X_{ideal}$ (Fig. 3(c)(d)). Conversely, $DWT^{-1}(L(X_{over}), H(X_{under}))$ exhibits a exposure similar to that of $X_{over}$, as shown in Fig. 3(b). To further validate these findings, we conducted a similar analysis using t-SNE dimensionality reduction [33] on 200 sample images in Fig. 3(e). The t-SNE results reveal that the high frequency components remain relatively consistent across different exposures, while the low frequency components exhibit significant variation.

Based on the above observations, we deduce that the ***low-frequency components primarily represent an image's lightness***, while the ***high-frequency components indicate structural details and are less affected by lightness variations***. By exploiting this characteristic, we can decompose the exposure of a distorted input image into two frequency representations and then construct an ideal reference image in the frequency space. Subsequently, differences in the frequency domain are mapped to pixel-level IEA results. This approach effectively mitigates any influence from irrelevant image semantics or noise associated with IEA.

**Pipeline.** P-IEANet comprises three essential modules (Fig. 4). The Structure Feature Module (*cf.* Sec. 3.1) and Lightness Feature Module (*cf.* Sec. 3.2) are responsible for extracting structural features from high-frequency components and lightness features from low-frequency components, respectively. Ultimately, the Prediction Module (*cf.* Sec. 3.3) integrates these features to predict pixel-level IEA results and generate the final prediction.

## 3.1 Structure Feature Module

There are two categories of structural features relevant to the IEA tasks: 1) Long-range features encompassing the overall layout and distant objects of an image, which provide a comprehensive understanding of its structure and global exposure [34–38]. 2) Short-range features focusing on fine details and textures, such as edges and localized patterns, are crucial for capturing local exposure variations [38–41]. To obtain these features effectively, we first derive gradient maps from the high-frequency components of the Haar DWT. These maps highlight edge regions, thus enhancing the representation of basic structural details [30, 38, 42]. Subsequently, we refine the extraction process by subjecting these gradient maps to a Long-Range Encoder (LRE) and a Short-Range Encoder (SRE). Further details are provided below.

**Gradient Maps.** The input image $X$ is decomposed by the Haar DWT to obtain high-frequency components, and then processed by a multi-layer encoder to extract naive features. For each layer $z_i$, where $i$ ranges from 1 to $N$ (the total number of layers), we compute the gradient map as follows:

$$\nabla z_i = \{g_d(z_i)|d \in D\}, \tag{2}$$

where $g_d(z_i)$ applies the first-order gradient function $g$ to $z_i$ in direction $d$. The set $D$ includes all directions under consideration: $+x, -x, +y, -y, +x+y, +x-y, -x+y, -x-y$, which correspond to the x-axis, y-axis, and their diagonals. These directions ensure comprehensive emphasis on edges, thereby enhancing the formulation of structural features.

**Long-range and Short-range Encoders.** To enhance the extraction of structural features, we feed both the original input feature, $z_i$, and its gradient maps $\nabla z_i$ into two distinct modules: a Transformer-based LRE $Z^l$ and a CNN-based SRE $Z^s$. The feature extraction process is as follows:

$$l_i = Z_i^l(z_i), \quad s_i = Z_i^s(z_i), \quad \nabla l_i = \nabla Z_i^l(\nabla z_i), \quad \nabla s_i = \nabla Z_i^s(\nabla z_i), \tag{3}$$

where $l_i$ and $s_i$ represent the long-range and short-range features, respectively, these features are then integrated using a Structure Fusion Module $Z^f$, which employs multiple MLPs, as follows:

$$Z_o = \left\| Z_i^f(l_i, s_i), Z_i^f(\nabla l_i, \nabla s_i) \right\|_i^N. \tag{4}$$

In this formulation, $\|...\|$ signifies the stacking of operations along the feature channel dimensions, facilitating a comprehensive synthesis of the extracted features.

## 3.2 Lightness Feature Module

The human visual system, possessing a high dynamic range, is skilled at globally detecting varying light levels of objects. However, due to limited attention capacity, it also ***tends to focus on specific***

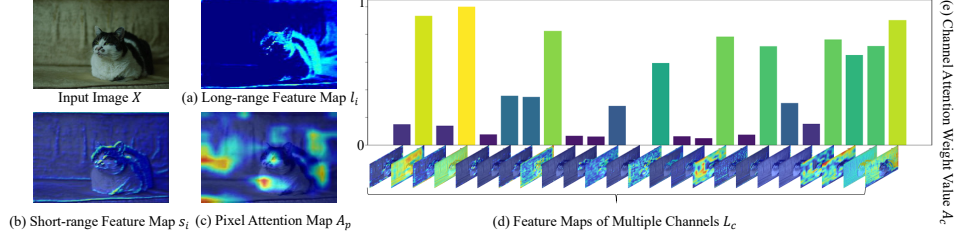

Figure 5: Visualization of different components in P-IEANet. Long-range (a) and Short-range (b) features highlight the important structural information for IEA tasks, while pixel-level attention (c) and channel-based attention (d-e) characterize its distribution in terms of light information.

*regions with distinct lightness levels*. Our method selectively processes global channels and local pixel regions to enhance the network's management of a broad spectrum of information.

**Lightness Channel Attention.** Firstly, our lightness channel attention (LCA) processes the channel-wise global spatial information, $L_c$ ($C \times H \times W$), through global average pooling to create a channel descriptor, $A_c$ ($C \times 1 \times 1$). To determine the weights for different channels, the descriptor undergoes further refinement in two convolution layers, followed by sigmoid $\sigma$ and ReLU $\gamma$ activation functions. This procedure is formalized as follows:

$$A_c = \sigma(\text{MLP}(\gamma(\text{MLP}(\frac{1}{H \times W} \sum_{i=1}^{H} \sum_{i=1}^{W} L_c(i,j)))))), \tag{5}$$

where $L_c(i,j)$ represents the lightness value at position $(i,j)$ in the $c$-th channel $L_c$, this channel attention strategy highlights that lightness variations across different channels convey distinct and weighted information. Finally, the channel weights are element-wise multiplied with the input to generate the output $F_c = A_c \otimes L_c$ ($C \times W \times H$).

**Lightness Pixel Attention.** The variable distribution of lightness among image pixels necessitates our lightness pixel attention (LPA) mechanism. This mechanism processes the output $F_c$ from the LCA using self-attention (SA) and convolution layers, coupled with ReLu and sigmoid activation functions. The attention mechanism is formulated as follows:

$$A_p = \sigma \left( \sum_{k \in K} w_k \cdot \text{Conv}_k(\gamma(\text{SA}(F_c))) \right). \tag{6}$$

Here, $\text{Conv}_k$ denotes a convolution operation with multi-scale kernel sizes $k$, which aims to enhance the network's focus on fine-grained and multi-scale exposure features under complex scenario. Larger kernel sizes help perceive overall brightness and contrast, while smaller kernel sizes detect localized overexposure or underexposure issues. Additionally, self-attention allows the system to analyze lightness distribution and recognize patterns at multiple scales.

We then perform element-wise multiplication to merge the input $L_c$ with $A_p$ ($1 \times H \times W$), generating the output $F_p = L_c \otimes A_p$ ($C \times H \times W$). The final stage integrates the outputs from both channel and pixel attention mechanisms to yield the comprehensive output $A_o = \text{MLP}(F_c, F_p)$.

### 3.3 Prediction Module

We employed Haar $\text{DWT}^{-1}$ to integrate lightness and structure features for ideal exposure representation reconstruction in the frequency space, then using the formula $P_p = \text{MLP}(DWT^{-1}(A_o, Z_o))$ to predict the *exposure residual, which measures the deviation of each pixel from the ideal exposure.* Both Haar DWT and $\text{DWT}^{-1}$ involve four dedicated convolutional kernels to simulate the wavelet transform's decomposition and reconstruction processes (Fig. 4(d)). To evaluate prediction accuracy, we define a loss function as:

$$\mathcal{L}_{pixel} = \frac{1}{H \times W} \sum_{i=1}^{H} \sum_{j=1}^{W} \left| P_p(i,j) - \widetilde{P}_p(i,j) \right|, \tag{7}$$

where $\widetilde{P}_p$ is the pixel-level ground truth, these residual maps can be converted into a coarser-grained prediction above pixel, e.g., holistic IEA score (*cf.* Sec. 5.2).

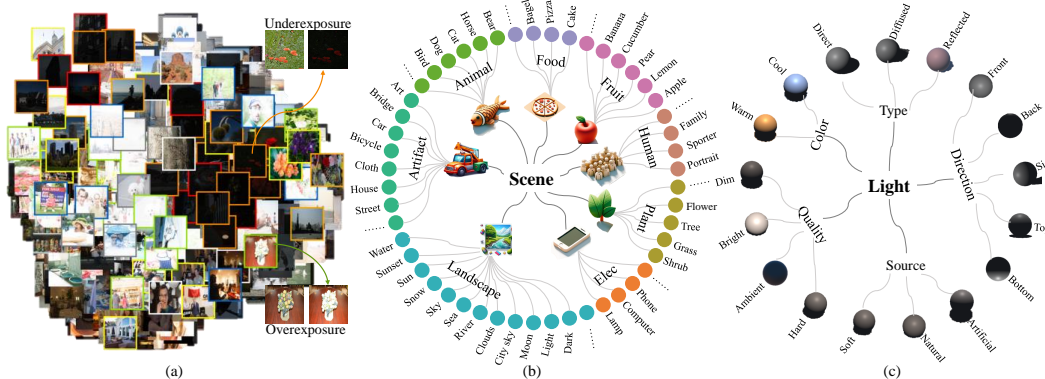

Figure 6: Proposed IEA40K dataset. (a) Visualization of images with different exposure conditions; (b) Scenes containing 8 super-classes with 50+ sub-classes; (c) 17 typical lighting scenarios.

## 4 Proposed IEA40K Dataset

### 4.1 Image Collection

***How to Avoid Selection Bias when Building a Comprehensive IEA Dataset.*** Selection bias can arise when certain types of intended exposure conditions, scenes and devices are underrepresented in the dataset, which may compromise the validity and generalizability of training models. To mitigate this issue, we considered 5 key aspects: varied scenes, diverse light conditions, sufficient devices, uniform resolution and comprehensive simulations (Fig. 6). These factors can significantly impact the quality of the dataset. Further details are provided in Appendix A.1.

***How to Align a set of Images through Pre-processing Strategies.*** Effectively aligning a set of images captured under various exposure conditions is a significant challenge. Factors like camera shake and slight subject movement during shooting parameter adjustments can lead to misalignment [43]. Such misalignment adversely affects the generation of supervised information and the training process. To tackle this issue, we first apply the Structural Similarity Index Measure (SSIM) algorithm to filter out misalignment images from the series. Those falling below a specified SSIM threshold are eliminated. Subsequently, we employ image alignment algorithms [43] to further enhance the alignment of the remaining images. This entire process is automated and can be executed in an unsupervised manner.

### 4.2 Data Annotation

***How to Obtain an Ideal Reference.*** Obtaining a reference image with ideal exposure in each region is crucial for our subsequent image annotation. However, ***ensuring the representatives of the reference image while minimizing the randomness and subjectivity introduced by humans presents significant challenges***. To address this, we start by creating a preliminary reference image using a multi-exposure fusion algorithm. Subsequently, we segment this image into blocks with the super-pixel segmentation algorithm [44] based on lightness and structure. Finally, experts optimize each block's exposure conditions utilizing ***Adams' zone system theory*** of classical photography [14, 15] (Fig. 1(c)). This theory provides precise guidelines for achieving ideal exposure across different elements.

***How to Obtain Pixel-Level Labels by Human-in-the-Loop Methods.*** Given the exorbitant cost and intricate nature of pixel-level annotation, we have streamlined the process using a combination of expert judgment and weak supervision techniques (Fig. 7). The ***initial*** pixel-level annotations were generated by comparing a reference image with the 8 distorted images, documenting ***exposure residual*** across pixels. Subsequently, experts further refined the ***final*** pixel-level annotations to rectify potential errors, such as accurately identifying areas with logos as severely overexposed and addressing discrete anomalous pixel labeling, to ensure that the final exposure residual closely aligns with the perceived deviation of each pixel from ideal exposure. For experts, distinguishing between the reference and distorted images is relatively straightforward and far more accurate, thus facilitating practical data annotation.

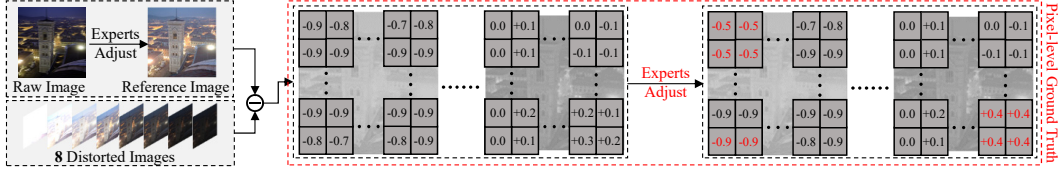

Figure 7: Overview of the proposed IEA dataset annotation process. First, collect images of different exposure conditions and then adjust the raw image by experts to obtain reference image; Second, calculate exposure residual between the reference and 8 distorted images, and then have experts adjust the local exposure residual. The closer it is to -1, the more overexposed the corresponding pixel is; and the closer it is to 1, the more underexposed the corresponding pixel is.

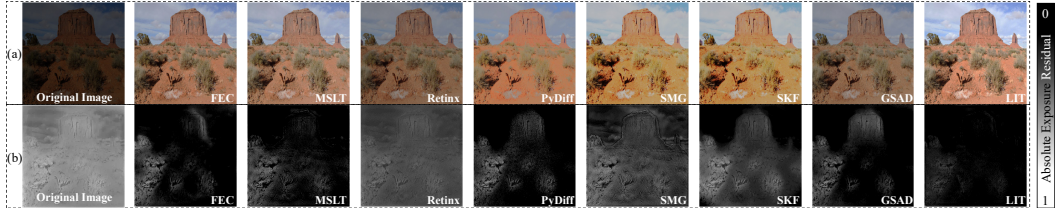

Figure 8: Example exposure residuals, a robust quantitative indicator for precise assessment, uniquely generated by P-IEANet (a lower absolute exposure residual suggests a visually more pleasing result). (a) Inputs: an original image and its enhanced counterparts by classical light enhancement methods. (b) Outputs: the exposure residuals hilighting the disparity between the input and the ideal exposure.

# 5 Experiments

## 5.1 Settings

**Benchmark Models and Training Protocols.** To the best of our knowledge, there is no publicly available pixel-level IEA model. Therefore, we have selected several deep learning baselines based on the following criteria: classical architectures with publicly available code and SOTA performance in a specific domain. For pixel-level IEA, we selected light enhancement, light-aware, and image quality assessment (IQA) models as backbones, complemented by appropriate output heads. For holistic IEA, we chose IQA and image aesthetics assessment (IAA) models to regress scores. Further details regarding training protocols can be found in Appendix A.3.

**Evaluation Metrics.** For pixel-level IEA, we adopt SSIM and MAE to measure the structure and lightness similarity between the ground truth and predicted exposure residual. For holistic IEA, we adopt the Spearman's rank correlation coefficient (SRCC) and the linear correlation coefficient (LCC), to measure the correlation between the predicted IEA score and human opinion [45].

## 5.2 Performance Evaluations

**Pixel-level Assessment.** Table 1 presents the results of P-IEANet and 12 other models on the IEA40k dataset. Our P-IEANet achieves SOTA performance, surpassing the second-best model with a remarkable -40% reduction in MAE loss and a significant +25% improvement in SSIM, while using an impressive -97% fewer training parameters. The efficiency of P-IEANet can be largely attributed to Haar DWT, which effectively minimizes the number of required feature extraction layers. Additionally, both the Lightness Feature Module and Structure Feature Module effectively utilize information, contributing to its exceptional performance with fewer parameters.

**Holistic Level Assessment.** We validated the effectiveness of P-IEANet on the holistic IEA task using the representative SPAQ dataset [12]. This dataset provides annotations for holistic exposure, allowing us to test the capabilities of various baseline models. P-IEANet supports three prediction methods: *1) With Fine-tuning:* after obtaining the residual map, it is processed through additional MLPs to predict the holistic exposure score, supervised by the MAE loss. A comparison of P-IEANet with 18 other models is presented in Table 2, where P-IEANet achieves SOTA performance. *2) Without Fine-tuning:* after obtaining the residual map, we compute the average of absolute values and subtract this average from 1, mapping it to a scale of 0-10 (Fig. 2). Remarkably, without requiring fine-tuning for holistic exposure scoring on SPAQ, we achieved a LRCC of 0.69 and SRCC of 0.65,

Table 1: Comparing the pixel-level IEA performance of 13 models on IEA40K. We adjusted the output headers of all light enhancement/awareness and IQA/IAA models to ensure their support for fine-tuning on our IEA40k. All models utilized only the exposure residual as pixel-level ground truth and **were retrained for the best performance.**

| Pixel level | Light Enhancement / Awareness | | | | | | | | IQA / IAA | | | | Ours |
|---|---|---|---|---|---|---|---|---|---|---|---|---|---|
| | FEC [32] | MSLT [46] | Retinex [47] | PyDiff [48] | SMG [38] | SKF [49] | GSAD [50] | LIT [51] | ArnIQA [52] | ReIQA [53] | DEIQT [54] | MUSIQ [55] | |
| Params | 9M | 8M | 5M | 374M | 575M | 22M | 67M | 146M | 106M | 560M | 363.5M | 298M | **2.7M** |
| MAE ↓ | 0.08 | 0.10 | 0.10 | 0.07 | 0.09 | 0.07 | 0.08 | 0.05 | 0.11 | 0.08 | 0.08 | 0.15 | **0.03** |
| SSIM ↑ | 0.35 | 0.33 | 0.42 | 0.50 | 0.46 | 0.44 | 0.41 | 0.60 | 0.50 | 0.37 | 0.39 | 0.24 | **0.75** |

Table 2: Comparing the holistic level IEA performance of 19 models on SPAQ. All models utilized only the exposure score as holistic-level ground truth, and **were retrained for the best performance.**

| Holistic level | Light Enhancement / Awareness | | | | | | | | IQA / IAA | | | | | | | | | | Ours |
|---|---|---|---|---|---|---|---|---|---|---|---|---|---|---|---|---|---|---|---|
| | FEC [32] | MSLT [46] | Retinex [47] | PyDiff [48] | SMG [38] | SKF [49] | GSAD [50] | LIT [51] | ArnIQA [52] | ReIQA [53] | DEIQT [54] | MUSIQ [55] | NIMA [45] | Alamp [56] | MLSP [57] | TANet [58] | EAT [59] | Q-align [60] | |
| Params | 9M | 8M | 5M | 374M | 575M | 22M | 67M | 146M | 106M | 560M | 363.5M | 298M | 56M | 99M | 24M | 40M | 87M | 7B | 2.7M |
| LRCC↑ | 0.74 | 0.75 | 0.71 | 0.70 | 0.69 | 0.68 | 0.71 | 0.72 | 0.70 | 0.71 | 0.63 | 0.64 | 0.63 | 0.67 | 0.71 | 0.75 | 0.74 | 0.75 | **0.78** |
| SRCC↑ | 0.69 | 0.70 | 0.67 | 0.68 | 0.67 | 0.66 | 0.68 | 0.69 | 0.68 | 0.65 | 0.54 | 0.60 | 0.60 | 0.64 | 0.68 | 0.69 | 0.71 | 0.70 | **0.73** |

*even surpassing some methods that do require fine-tuning.* The above results show that P-IEANet exhibits strong criteria and scenario robustness beyond pixel-level tasks. *3) Criteria-oriented without Fine-tuning:* Moreover, we additionally discuss an *industry-applicable criteria-oriented scoring methodology* in Appendix A.2.

**Ablation Studies.** Table 3 evaluates the effectiveness of P-IEANet's modules. The absence of the Haar DWT and two other modules, Structure and Lightness, significantly impact the performance of P-IEANet. Specifically, the SSIM decreases by 42.6%, 28.0%, and 44.0% respectively, while the SRCC falls by 14.3%, 7.1%, and 15.8%. These results confirm that each module, particularly the Lightness Feature Module which processes low-frequency information, plays a crucial role in enhancing the model's overall performance. Qualitative visual effects analysis is provided in Fig. 5.

**Predictions for Images.** The prediction examples are shown in Fig. 2. Similar to human perception, P-IEANet's pixel-level evaluation results effectively identify areas of overexposure and underexposure that are visually displeasing, even in unconventional scenes where both underexposure and overexposure coexist. Moreover, when combined with semantic segmentation algorithms, P-IEANet enables more precise object- and pixel-level IEA results (Appendix A.2).

## 5.3 Advancing Light Enhancement Methods

P-IEANet, owing to its exceptional sensitivity towards exposure, offers advantages for the exposure enhancement community in the following two aspects:

*1) Analyzing Performance Better:* Traditionally, assessing the efficacy of image enhancement algorithms has been a time-consuming and imprecise task, relying solely on human observations. The exposure residual, uniquely generated by P-IEANet, serves as a robust quantitative indicator for precise assessment (refer to Fig. 8, where input can be either an original image or an enhanced one).

*2) Enhancing Performance Better:* P-IEANet is compatible with many existing light enhancement methods, enabling it to boost their performance. To demonstrate this, we chose two open-source and SOTA methods, Retinex [47] and GASD [50], as baseline models. We incorporated P-IEANet as a sample evaluator in these models after enhancement, freezing P-IEANet's parameters and obtaining the absolute exposure residual as loss to include in the baseline models. Table 4 shows that on both representative datasets, LOL-v1 [61] and LOLv2-real [62], the performance is improved to some extent, suggesting that P-IEANet has the potential to become an important enhancer in this field.

Table 3: Ablation studies conducted on IEA40K.

| Method | Pixel-level | | | Holistic level | | |
|---|---|---|---|---|---|---|
| | Params | MAE | SSIM | ACC | LRCC | SRCC |
| w/o DWT | 2.7M | 0.08 | 0.43 | 0.81 | 0.74 | 0.72 |
| w/o Structure | 1.6M | 0.05 | 0.54 | 0.84 | 0.81 | 0.78 |
| w/o Lightness | 1.0M | 0.08 | 0.42 | 0.73 | 0.74 | 0.71 |
| P-IEANet | 2.7M | **0.03** | **0.75** | **0.88** | **0.89** | **0.84** |

Table 4: Our P-IEANet can enhance some low-light enhancement methods. Our retrained results are marked by '*'.

| Model | LOL-v1 | | LOLv2-real | |
|---|---|---|---|---|
| | PSNR↑ | SSIM↑ | PSNR↑ | SSIM↑ |
| Retinex (ICCV'23) | 25.16 | 0.845 | 22.80 | 0.840 |
| Retinex* | 24.7 | 0.80 | 21.9 | 0.82 |
| Retinex* + P-IEANet | **25.3** | **0.85** | **23.0** | **0.86** |
| GASD (NIPS'23) | 27.839 | 0.877 | 28.818 | 0.895 |
| GASD* | 27.1 | 0.85 | 28.4 | 0.86 |
| GASD* + P-IEANet | **28.2** | **0.88** | **29.5** | **0.91** |

## 6 Conclusion

This paper investigates IEA with a novel paradigm: from holism to pixel. To our knowledge, our work introduces a new roadmap by proposing a model, dataset, and benchmark for the community. However, several challenges still remain to be addressed. For instance, evaluating images with severe misalignment issues caused by high-speed moving objects poses significant challenges. In future work, we aim to optimize our framework to support multimodal outputs and enhance the exposure perception in artificial intelligence generated content (AIGC).

## 7 Acknowledgement

This work is supported by the Funds for Creative Research Groups of China under Grant 61921003.

## Footnotes

† Equal contribution. * Corresponding author.

## References

[1] Shaoqing Ren, Kaiming He, Ross Girshick, and Jian Sun. Faster r-cnn: Towards real-time object detection with region proposal networks. *Advances in neural information processing systems*, 28, 2015.

[2] Jiankang Deng, Jia Guo, Niannan Xue, and Stefanos Zafeiriou. Arcface: Additive angular margin loss for deep face recognition. In *Proceedings of the IEEE/CVF conference on computer vision and pattern recognition*, pages 4690–4699, 2019.

[3] Wenhan Yang, Ye Yuan, Wenqi Ren, Jiaying Liu, Walter J Scheirer, Zhangyang Wang, Taiheng Zhang, Qiaoyong Zhong, Di Xie, Shiliang Pu, et al. Advancing image understanding in poor visibility environments: A collective benchmark study. *IEEE Transactions on Image Processing*, 29:5737–5752, 2020.

[4] Ritendra Datta, Dhiraj Joshi, Jia Li, and James Z Wang. Studying aesthetics in photographic images using a computational approach. In *European conference on computer vision*, pages 288–301. Springer, 2006.

[5] Min Liu, Po Yuan, and Richard S Turner Jr. Automatic analysis and adjustment of digital images with exposure problems, Apr. 15 2008. US Patent 7,359,572.

[6] Juan Torres and José Manuel Menéndez. Optimal camera exposure for video surveillance systems by predictive control of shutter speed, aperture, and gain. In *Real-Time Image and Video Processing 2015*, volume 9400, pages 238–251. SPIE, 2015.

[7] Madasu Hanmandlu, Om Prakash Verma, Nukala Krishna Kumar, and Muralidhar Kulkarni. A novel optimal fuzzy system for color image enhancement using bacterial foraging. *IEEE Transactions on Instrumentation and Measurement*, 58(8):2867–2879, 2009.

[8] Mohammad T Rahman, Nasser Kehtarnavaz, and Qolamreza R Razlighi. Using image entropy maximum for auto exposure. *Journal of electronic imaging*, 20(1):013007, 2011.

[9] Huimin Lu, Hui Zhang, Shaowu Yang, and Zhiqiang Zheng. Camera parameters auto-adjusting technique for robust robot vision. In *2010 IEEE International Conference on Robotics and Automation*, pages 1518–1523. IEEE, 2010.

[10] Wufeng Xue, Xuanqin Mou, Lei Zhang, Alan C Bovik, and Xiangchu Feng. Blind image quality assessment using joint statistics of gradient magnitude and laplacian features. *IEEE Transactions on Image Processing*, 23(11):4850–4862, 2014.

[11] Anish Mittal, Anush Krishna Moorthy, and Alan Conrad Bovik. No-reference image quality assessment in the spatial domain. *IEEE Transactions on image processing*, 21(12):4695–4708, 2012.

[12] Yuming Fang, Hanwei Zhu, Yan Zeng, Kede Ma, and Zhou Wang. Perceptual quality assessment of smartphone photography. In *Proceedings of the IEEE/CVF Conference on Computer Vision and Pattern Recognition*, pages 3677–3686, 2020.

[13] Anuj Srivastava, Ann B Lee, Eero P Simoncelli, and S-C Zhu. On advances in statistical modeling of natural images. *Journal of mathematical imaging and vision*, 18:17–33, 2003.

[14] Ansel Adams. *Basic Photo: The Negative v. 2*. New York Graphic Society, New York, NY, Mar. 1978.

[15] Ansel Adams. *New photo series 2: Negative:*. Ansel Adams Photography. Bulfinch Press, New York, NY, June 1995.

[16] Inwook Shim, Joon-Young Lee, and In So Kweon. Auto-adjusting camera exposure for outdoor robotics using gradient information. In *2014 IEEE/RSJ International Conference on Intelligent Robots and Systems*, pages 1011–1017. IEEE, 2014.

[17] Inwook Shim, Tae-Hyun Oh, Joon-Young Lee, Jinwook Choi, Dong-Geol Choi, and In So Kweon. Gradient-based camera exposure control for outdoor mobile platforms. *IEEE Transactions on Circuits and Systems for Video Technology*, 29(6):1569–1583, 2018.

[18] Zichao Zhang, Christian Forster, and Davide Scaramuzza. Active exposure control for robust visual odometry in hdr environments. In *2017 IEEE international conference on robotics and automation (ICRA)*, pages 3894–3901. IEEE, 2017.

[19] Joowan Kim, Younggun Cho, and Ayoung Kim. Exposure control using bayesian optimization based on entropy weighted image gradient. In *2018 IEEE International conference on robotics and automation (ICRA)*, pages 857–864. IEEE, 2018.

[20] S Efimov, A Nefyodov, and M Rychagov. Block-based image exposure assessment and indoor/outdoor classification. In *Proc. of 17th Conf. on Computer Graphics GraphiCon*, 2007.

[21] Xuan Dong, Lu Yuan, Weixin Li, and Alan L Yuille. Temporally consistent region-based video exposure correction. In *2015 IEEE International Conference on Multimedia and Expo (ICME)*, pages 1–6. IEEE, 2015.

[22] Yuzhe Yang, Liwu Xu, Leida Li, Nan Qie, Yaqian Li, Peng Zhang, and Yandong Guo. Personalized image aesthetics assessment with rich attributes. *arXiv preprint arXiv:2203.16754*, 2022.

[23] Shu Kong, Xiaohui Shen, Zhe Lin, Radomir Mech, and Charless Fowlkes. Photo aesthetics ranking network with attributes and content adaptation. In *European conference on computer vision*, pages 662–679. Springer, 2016.

[24] Wenhan Zhu, Guangtao Zhai, Zongxi Han, Xiongkuo Min, Tao Wang, Zicheng Zhang, and Xiaokang Yangand. A multiple attributes image quality database for smartphone camera photo quality assessment. In *2020 IEEE International Conference on Image Processing (ICIP)*, pages 2990–2994. IEEE, 2020.

[25] Michele A Saad, Alan C Bovik, and Christophe Charrier. Blind image quality assessment: A natural scene statistics approach in the dct domain. *IEEE transactions on Image Processing*, 21(8):3339–3352, 2012.

[26] Lijun Zhang, Lin Zhang, Xiao Liu, Ying Shen, and Dongqing Wang. Image exposure assessment: a benchmark and a deep convolutional neural networks based model. In *2018 IEEE International Conference on Multimedia and Expo (ICME)*, pages 1–6. IEEE, 2018.

[27] Ansel Adams. The negative: Exposure and development. ansel adams basic photography series/book 2. *New York Graphic Society, Boston, MA, USA*, 1948.

[28] Weijia Wu, Yuzhong Zhao, Mike Zheng Shou, Hong Zhou, and Chunhua Shen. Diffumask: Synthesizing images with pixel-level annotations for semantic segmentation using diffusion models. In *Proceedings of the IEEE/CVF International Conference on Computer Vision*, pages 1206–1217, 2023.

[29] Zhongwei Ren, Zhicheng Huang, Yunchao Wei, Yao Zhao, Dongmei Fu, Jiashi Feng, and Xiaojie Jin. Pixellm: Pixel reasoning with large multimodal model. *arXiv preprint arXiv:2312.02228*, 2023.

[30] Anmin Liu, Weisi Lin, and Manish Narwaria. Image quality assessment based on gradient similarity. *IEEE Transactions on Image Processing*, 21(4):1500–1512, 2011.

[31] Dario Fuoli, Luc Van Gool, and Radu Timofte. Fourier space losses for efficient perceptual image super-resolution. In *Proceedings of the IEEE/CVF International Conference on Computer Vision*, pages 2360–2369, 2021.

[32] Jie Huang, Yajing Liu, Feng Zhao, Keyu Yan, Jinghao Zhang, Yukun Huang, Man Zhou, and Zhiwei Xiong. Deep fourier-based exposure correction network with spatial-frequency interaction. In *European Conference on Computer Vision*, pages 163–180. Springer, 2022.

[33] Laurens Van der Maaten and Geoffrey Hinton. Visualizing data using t-sne. *Journal of machine learning research*, 9(11), 2008.

[34] Hanting Chen, Yunhe Wang, Tianyu Guo, Chang Xu, Yiping Deng, Zhenhua Liu, Siwei Ma, Chunjing Xu, Chao Xu, and Wen Gao. Pre-trained image processing transformer. In *Proceedings of the IEEE/CVF conference on computer vision and pattern recognition*, pages 12299–12310, 2021.

[35] Ze Liu, Yutong Lin, Yue Cao, Han Hu, Yixuan Wei, Zheng Zhang, Stephen Lin, and Baining Guo. Swin transformer: Hierarchical vision transformer using shifted windows. In *Proceedings of the IEEE/CVF international conference on computer vision*, pages 10012–10022, 2021.

[36] Haiping Wu, Bin Xiao, Noel Codella, Mengchen Liu, Xiyang Dai, Lu Yuan, and Lei Zhang. Cvt: Introducing convolutions to vision transformers. In *Proceedings of the IEEE/CVF international conference on computer vision*, pages 22–31, 2021.

[37] Kun Yuan, Shaopeng Guo, Ziwei Liu, Aojun Zhou, Fengwei Yu, and Wei Wu. Incorporating convolution designs into visual transformers. In *Proceedings of the IEEE/CVF international conference on computer vision*, pages 579–588, 2021.

[38] Xiaogang Xu, Ruixing Wang, and Jiangbo Lu. Low-light image enhancement via structure modeling and guidance. In *Proceedings of the IEEE/CVF Conference on Computer Vision and Pattern Recognition*, pages 9893–9903, 2023.

[39] Elad Richardson, Yuval Alaluf, Or Patashnik, Yotam Nitzan, Yaniv Azar, Stav Shapiro, and Daniel Cohen-Or. Encoding in style: a stylegan encoder for image-to-image translation. In *Proceedings of the IEEE/CVF conference on computer vision and pattern recognition*, pages 2287–2296, 2021.

[40] Omer Tov, Yuval Alaluf, Yotam Nitzan, Or Patashnik, and Daniel Cohen-Or. Designing an encoder for stylegan image manipulation. *ACM Transactions on Graphics (TOG)*, 40(4):1–14, 2021.

[41] Tengfei Wang, Yong Zhang, Yanbo Fan, Jue Wang, and Qifeng Chen. High-fidelity gan inversion for image attribute editing. In *Proceedings of the IEEE/CVF Conference on Computer Vision and Pattern Recognition*, pages 11379–11388, 2022.

[42] Wenqi Ren, Sifei Liu, Lin Ma, Qianqian Xu, Xiangyu Xu, Xiaochun Cao, Junping Du, and Ming-Hsuan Yang. Low-light image enhancement via a deep hybrid network. *IEEE Transactions on Image Processing*, 28(9):4364–4375, 2019.

[43] Jirong Zhang, Chuan Wang, Shuaicheng Liu, Lanpeng Jia, Nianjin Ye, Jue Wang, Ji Zhou, and Jian Sun. Content-aware unsupervised deep homography estimation. In *Computer Vision–ECCV 2020: 16th European Conference, Glasgow, UK, August 23–28, 2020, Proceedings, Part I 16*, pages 653–669. Springer, 2020.

[44] Radhakrishna Achanta, Appu Shaji, Kevin Smith, Aurelien Lucchi, Pascal Fua, and Sabine Süsstrunk. Slic superpixels compared to state-of-the-art superpixel methods. *IEEE transactions on pattern analysis and machine intelligence*, 34(11):2274–2282, 2012.

[45] Hossein Talebi and Peyman Milanfar. Nima: Neural image assessment. *IEEE transactions on image processing*, 27(8):3998–4011, 2018.

[46] Yijie Zhou, Chao Li, Jin Liang, Tianyi Xu, Xin Liu, and Jun Xu. 4k-resolution photo exposure correction at 125 fps with˜ 8k parameters. In *Proceedings of the IEEE/CVF Winter Conference on Applications of Computer Vision*, pages 1587–1597, 2024.

[47] Yuanhao Cai, Hao Bian, Jing Lin, Haoqian Wang, Radu Timofte, and Yulun Zhang. Retinexformer: One-stage retinex-based transformer for low-light image enhancement. In *Proceedings of the IEEE/CVF International Conference on Computer Vision*, pages 12504–12513, 2023.

[48] Dewei Zhou, Zongxin Yang, and Yi Yang. Pyramid diffusion models for low-light image enhancement. *arXiv preprint arXiv:2305.10028*, 2023.

[49] Yuhui Wu, Chen Pan, Guoqing Wang, Yang Yang, Jiwei Wei, Chongyi Li, and Heng Tao Shen. Learning semantic-aware knowledge guidance for low-light image enhancement. In *Proceedings of the IEEE/CVF Conference on Computer Vision and Pattern Recognition*, pages 1662–1671, 2023.

[50] Jinhui Hou, Zhiyu Zhu, Junhui Hou, Hui Liu, Huanqiang Zeng, and Hui Yuan. Global structure-aware diffusion process for low-light image enhancement. *Advances in Neural Information Processing Systems*, 36, 2024.

[51] Zhexin Liang, Chongyi Li, Shangchen Zhou, Ruicheng Feng, and Chen Change Loy. Iterative prompt learning for unsupervised backlit image enhancement. In *Proceedings of the IEEE/CVF International Conference on Computer Vision*, pages 8094–8103, 2023.

[52] Lorenzo Agnolucci, Leonardo Galteri, Marco Bertini, and Alberto Del Bimbo. Arniqa: Learning distortion manifold for image quality assessment. In *Proceedings of the IEEE/CVF Winter Conference on Applications of Computer Vision*, pages 189–198, 2024.

[53] Avinab Saha, Sandeep Mishra, and Alan C Bovik. Re-iqa: Unsupervised learning for image quality assessment in the wild. In *Proceedings of the IEEE/CVF Conference on Computer Vision and Pattern Recognition*, pages 5846–5855, 2023.

[54] Guanyi Qin, Runze Hu, Yutao Liu, Xiawu Zheng, Haotian Liu, Xiu Li, and Yan Zhang. Data-efficient image quality assessment with attention-panel decoder. In *Proceedings of the AAAI Conference on Artificial Intelligence*, volume 37, pages 2091–2100, 2023.

[55] Junjie Ke, Qifei Wang, Yilin Wang, Peyman Milanfar, and Feng Yang. Musiq: Multi-scale image quality transformer. In *Proceedings of the IEEE/CVF international conference on computer vision*, pages 5148–5157, 2021.

[56] Shuang Ma, Jing Liu, and Chang Wen Chen. A-lamp: Adaptive layout-aware multi-patch deep convolutional neural network for photo aesthetic assessment. In *CVPR*, pages 722–731, 2017.

[57] Vlad Hosu, Bastian Goldlucke, and Dietmar Saupe. Effective aesthetics prediction with multi-level spatially pooled features. In *CVPR*, 2019.

[58] Shuai He, Yongchang Zhang, Rui Xie, Dongxiang Jiang, and Anlong Ming. Rethinking image aesthetics assessment: Models, datasets and benchmarks. *IJCAI*, 2022.

[59] Shuai He, Anlong Ming, Shuntian Zheng, Haobin Zhong, and Huadong Ma. Eat: An enhancer for aesthetics-oriented transformers. In *Proceedings of the 31st ACM International Conference on Multimedia*, pages 1023–1032, 2023.

[60] Haoning Wu, Zicheng Zhang, Weixia Zhang, Chaofeng Chen, Liang Liao, Chunyi Li, Yixuan Gao, Annan Wang, Erli Zhang, Wenxiu Sun, et al. Q-align: Teaching lmms for visual scoring via discrete text-defined levels. *arXiv preprint arXiv:2312.17090*, 2023.

[61] Chen Wei, Wenjing Wang, Wenhan Yang, and Jiaying Liu. Deep retinex decomposition for low-light enhancement. *arXiv preprint arXiv:1808.04560*, 2018.

[62] Wenhan Yang, Wenjing Wang, Haofeng Huang, Shiqi Wang, and Jiaying Liu. Sparse gradient regularized deep retinex network for robust low-light image enhancement. *IEEE Transactions on Image Processing*, 30:2072–2086, 2021.

[63] Vladimir Bychkovsky, Sylvain Paris, Eric Chan, and Frédo Durand. Learning photographic global tonal adjustment with a database of input/output image pairs. In *CVPR 2011*, pages 97–104. IEEE, 2011.

[64] Diederik P Kingma and Jimmy Ba. Adam: A method for stochastic optimization. *arXiv preprint arXiv:1412.6980*, 2014.

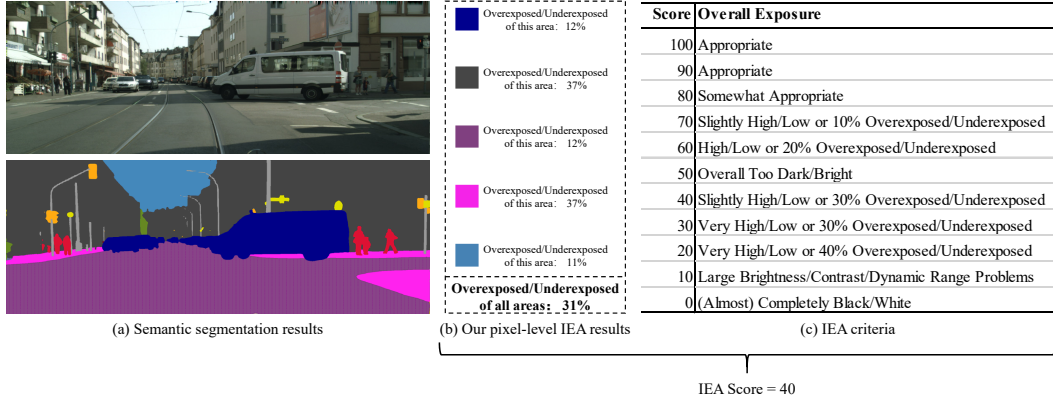

(a) Semantic segmentation results    (b) Our pixel-level IEA results    (c) IEA criteria

IEA Score = 40

Figure 9: Based on the IEA criteria provided by different manufacturers, our pixel-level IEA results can be mapped to the corresponding IEA scores according to the provided criteria without fine-tuning.

## A Appendix

### A.1 Image Collection Details

*1) Varied Scenes.* The scene defines the image's content and overall lighting conditions. As shown in Fig. 6(a)(b), IEA40K spans over 50 scene categories, enabling a comprehensive understanding of image exposure in dynamic scenes. *2) Diverse Light.* The type and direction of the light source greatly influence light distribution within an image. Our dataset includes a variety of light sources and directions, as shown in Fig. 6(c). We also consider the shading tone's impact on an image's visual appeal, selecting images with dark, clear, and neutral tones. *3) Sufficient Devices.* The performance of the capturing device can affect image quality and exposure. To address this, our dataset features images from over 27 different devices, ranging from smartphones like the iPhone 13 and Huawei Mate40 to digital cameras from Canon, Nikon, and others. *4) Uniform Resolution.* Initially captured at high resolutions, our images were downsized for manageability. Keeping the aspect ratio intact, we reduced the image size so that the shorter side measures 512 pixels, storing them in PNG format to balance quality and model compatibility. *5) Comprehensive Simulations.* Capturing varied exposure levels for the same scene poses a challenge. To overcome this, we simulated 80% of the images at different exposure EV levels from the original raw files, or collected images from MIT-Adobe FiveK dataset [63]. After expert selection and adjustment, we compiled a set of 8 specific exposure levels for each raw image, resulting in 1 ideal reference and 8 exposure distorted images per scene.

### A.2 Criteria-oriented IEA without fine-tuning

Different manufacturers have their own criteria for IEA scoring. Our pixel-level IEA results can adapt well to these different criteria. In Fig. 9, we provide an example where manufacturers can map the pixel-level IEA results to their overall scoring criteria to ultimately obtain their IEA scores.

### A.3 Experimental Settings

Our model is implemented in PyTorch and trained with the Adam optimizer [64]. We set the initial learning rate to $3 \times 10^{-5}$ with a decay rate of 0.1 after every 10 epochs, and the mini-batch size is set to 36. On the Intel 10900X CPU and RTX 3090 platform, the entire training time is about 14 hours for 30 epochs (early stopping), and the inference time is 0.083 seconds for a $256 \times 256$ image (supports larger input sizes). To reduce the bias caused by a random splitting, we run the random train-test splitting operation five times, and the comparison results are reported as the average of the five evaluation experiments.

### A.4 Why not Employ Light Enhancement Methods to Obtain an Exposure Residual?

There is a common misconception that subtracting the inputs and outputs from existing light enhancement methods, or comparing distorted images to reference images from existing light enhancement

datasets, will yield the pixel-level IEA prediction results or ground truth (exposure residual) described in this paper. However, there are two fundamental differences: ***1) Different Data Production Processes:*** light enhancement methods only optimize reference images' exposure in ***global*** areas through linear adjustments, e.g., adjusting the exposure value (EV) across the entire image. Our method involves non-linear adjustments by experts to optimize ***local*** exposure in specific areas of reference images. Additionally, human experts correct any errors in the exposure residual. As a result, the exposure residual obtained from these two approaches contains different amounts of information. ***2) Different Task Objectives:*** light enhancement methods generally aim to rectify image issues to a level acceptable to human vision, ensuring consistency in image semantics and using complete reference images for supervision. In contrast, our task is focused solely on resolving exposure issues. We do not consider the impact of image semantics on predictions nor attempt to restore image semantics. Instead, we use only the exposure residual for supervision, eliminating other factors that could impact exposure quality.

## A.5 More Performance Evaluations

**Comparison of Different Wavelets.** As outlined in the paper, the Haar wavelet distinctly aligns its component decomposition with exposure characteristics, a unique attribute not shared by other wavelets. Experimental results also show that the Haar wavelet surpasses other wavelets in performance. Table 5 presents a comparative analysis of the Haar wavelet against other notable wavelets (Daubechies and Symlet) on the IEA40K dataset.

Table 5: Ablation of various wavelets.

|  | MAE↓ | SSIM↑ |
|---|---|---|
| Haar | **0.03** | **0.75** |
| Daubechies | 0.07 | 0.49 |
| Symlet | 0.08 | 0.41 |

Table 6: Ablation of different loss functions.

|  | Epoch↓ | SSIM↑ |
|---|---|---|
| L1-norm | **30** | **0.75** |
| L2-norm | 50 | 0.62 |
| Smooth L1 | 45 | 0.67 |

**Comparison of Different Loss Functions.** During training, we evaluated three primary types of loss functions: L1-norm, L2-norm, and Smooth L1. Comparative results are detailed in Table 6. The L1-norm demonstrates superior robustness and faster training speeds, as evidenced by earlier convergence epochs in the IEA task.

Table 7: The PSNR results on IEA40k.

|  | FEC | MSLT | Retinex | SMG |
|---|---|---|---|---|
| PSNR↑ | 23.22 | 21.02 | 23.8 | 24.17 |
|  | SKF | GSAD | LIT | Ours |
| PSNR↑ | 21.68 | 19.37 | 23.51 | **28.04** |

**Comparison of PSNR Performance.** PSNR can also serve as an evaluation metric at the pixel level, and comparisons of some methods are presented in Table 7, which also demonstrates our SOTA performance.

## A.6 Limitations

The dataset includes few images of high-speed moving objects because capturing well-aligned images with different exposures for such objects is difficult. The misalignment cannot be easily corrected using existing image alignment methods. Therefore, different exposure images for these objects can only be simulated by adjusting the EV value of ideal images.

Additionally, resizing high-resolution images for evaluation can significantly compress the image, affecting pixel-level exposure assessments. Increasing the input size during network training can address this issue but will also increase the inference time.

## A.7 Safeguards and Licenses for Existing Assets

The original owners of assets (e.g., code, data, models) used in the paper are properly credited, and the licenses and terms of use are explicitly mentioned and properly respected, ensuring that there are no copyright issues and no risk of misuse.

